# Free energy score-space

**Alessandro Perina**[1,3], **Marco Cristani**[1,2], **Umberto Castellani**[1]
**Vittorio Murino**[1,2] and **Nebojsa Jojic**[3]
{alessandro.perina, marco.cristani, umberto.castellani, vittorio.murino}@univr.it
jojic@microsoft.com
[1] Department of Computer Science, University of Verona, Italy
[2] IIT, Italian Institute of Technology, Genova, Italy
[3] Microsoft Research, Redmond, WA

## Abstract

A score function induced by a generative model of the data can provide a feature vector of a fixed dimension for each data sample. Data samples themselves may be of differing lengths (e.g., speech segments, or other sequence data), but as a score function is based on the properties of the data generation process, it produces a fixed-length vector in a highly informative space, typically referred to as a "score space". Discriminative classifiers have been shown to achieve higher performance in appropriately chosen score spaces than is achievable by either the corresponding generative likelihood-based classifiers, or the discriminative classifiers using standard feature extractors. In this paper, we present a novel score space that exploits the free energy associated with a generative model. The resulting free energy score space (**FESS**) takes into account latent structure of the data at various levels, and can be trivially shown to lead to classification performance that at least matches the performance of the free energy classifier based on the same generative model, and the same factorization of the posterior. We also show that in several typical vision and computational biology applications the classifiers optimized in **FESS** outperform the corresponding pure generative approaches, as well as a number of previous approaches to combining discriminating and generative models.

## 1 Introduction

The complementary nature of discriminative and generative approaches to machine learning [20] has motivated lots of research on the ways in which these can be combined [5, 12, 15, 18, 9, 24, 27]. One recipe for such integration uses "generative score-spaces." Using the notation of [24], such spaces can be built from data by considering for each observed sequence $x = (x_1, \ldots, x_k, \ldots, x_K)$ of observations $x_k \in \Re^d$, $k = 1, \ldots, K$, a family of generative models $\mathcal{P} = \{P(x|\theta_i)\}$ parameterized by $\theta_i$.

The observed sequence $x$ is mapped to the fixed-length score vector $\varphi_{\hat{F}}^f(x)$,

$$\varphi_{\hat{F}}^f(x) = \varphi_{\hat{F}} f(\{P_i(x|\theta_i))\}), \qquad (1)$$

where $f$ is the function of the set of probability densities under the different models, and $\hat{F}$ is some operator applied to it. For instance, in case of the Fisher score [9], $f$ is the log likelihood, and the operator $\hat{F}$ produces the first order derivatives with respect to parameters, whereas in [24] other derivatives are also included. Another example is the TOP kernel [27] for which the function $f$ is the posterior log-odds and $\hat{F}$ is again the gradient operator.

In these cases, the generative score-space approaches help to distill the relationship between a model parameter $\theta_i$ and the particular data sample. After the mapping, a score-space metric must

be defined in order to employ discriminative approaches.

A number of nice properties for these mappings, and especially for Fisher score, can be derived under the assumption that the test data indeed follows the generative model used for the score computation. However, the generative score spaces build upon the choice of one (or few) out of many possible generative models, as well as the parameters fit to a limited amount of data. In practice, these models can therefore suffer from improper parametrization of the probability density function, local minima, over-fitting add under-training problems. Consider, for instance, the situation where the assumed model over high dimensional data is a mixture of $n$ diagonal Gaussians with a given small and fixed variance, and a uniform prior over the components. The only free parameters are therefore the Gaussian centers, and let us assume that training data is best captured with these centers all lying on (or close to) a hypersphere with a radius sufficiently larger than the Gaussians' deviation. An especially surprising and inconvenient outlier in this case would be a test data point that falls close to the center of the hypersphere, as the derivatives of its log likelihood with respect to these parameters (Gaussian centers) evaluated at the estimate could be very low when the number of components $n$ in the mixture is large, because the derivatives are scaled by the uniform posterior $1/n$. But, this makes such a test point insufficiently distinguishable from the test points that actually satisfy the model perfectly by falling directly into one of the Gaussian centers. If the model parameters are extended to include the prior distribution over mixture components, then derivatives with respect to these parameters would help disambiguate these points.

In this paper, we propose a novel score space which focuses on how well the data point fits different parts of the generative model, rather than on derivatives with respect to the model parameters. We start with the variational free energy as a lower bound on the negative log-likelihood of the data, as this affords us with two advantages. First of all, the variational free energy can be computed for an arbitrary structure of the posterior distribution, allowing us to deal with generative models with many latent variables and complex structure without compromising tractability, as was previously done for inference in generative models. Second, a variational approximation of the posterior typically provides an additive decomposition of the free energy, providing many terms that can be used as features. These terms/features are divided into two categories: the "entropy set" of terms that express uncertainty in the posterior distribution, and the "cross-entropy set" describing the quality of the fit of the data to different parts of the model according to the posterior distribution.

We find the resulting score space to be highly informative for discriminative learning. In particular, we tested our approach on three computational biology problems (promoter recognition, exons/introns classification, and homology detection), as well as vision problems (scene/object recognition). The results compare favorably with the state-of-the-art from recent literature.

The rest of the paper is organized as follows. The next section describes the proposed framework in more detail. In Sec. 3, we show that the proposed generative score space leads to better classification performances than the related generative counterpart. Some simple extensions are described in Sec. 4, and used in the experiments in Sec. 5.

## 2 FESS: Free Energy Score Space

A generative model defines the distribution $P(h, x|\theta) = \prod_{t=1}^{T} P(h^{(t)}, x^{(t)}|\theta)$ over a set of observations $x = \{x^{(t)}\}_{t=1}^{T}$, each with associated hidden variables $h^{(t)}$, for a given set of model parameters $\theta$ shared across all observations. In addition, to model the posterior distribution $P(h|x)$, we also define a family of distributions $\mathcal{Q}$ from which we need to select a variational distribution $Q(h)$ that best fits the model and the data. Assuming i.i.d data, the family $\mathcal{Q}$ can be simplified to include only distributions of the form $Q(h) = \prod_{t=1}^{T} q(h^{(t)})$. The *free energy* [19, 11] is a function of the data, parameters of the posterior $Q(h)$, and the parameters of the model $P$, defined as

$$\mathcal{F}_\mathcal{Q} = \mathbb{KL}(Q, P(h|x, \theta)) - \log P(x|\theta) = \sum_h Q(h) \log \frac{Q(h)}{P(h, x|\theta)} \qquad (2)$$

The free energy bounds the log likelihood, $\mathcal{F}_\mathcal{Q} \leq -\log P(x)$ and the equality is attained only if $\mathcal{Q}$ is expressive enough to capture the true posterior distribution, as the free energy is minimized when $Q(h) = P(h|x)$. Constraining $Q$ to belong to a simplified family of distributions $\mathcal{Q}$, however,

provides computational advantages for dealing with intractable models $P$. Examples of distribution families used for approximation are the fully-factorized mean field form [13], or the structured variational approximation [7], where some dependencies among the hidden variables are kept.

Minimization of $\mathcal{F}_\mathcal{Q}$ as a proxy for negative log likelihood is usually achieved by alternating optimization of with respect to $Q$ and $\theta$, a special case of which – when $Q$ is fully expressive – is the EM algorithm. Different choices of $\mathcal{Q}$ provide different types of compromise between the accuracy and computational complexity. For some models, accurate inference of some of the latent variables may require excessive computation even though the results of the inference can be correctly reinterpreted by studying the posterior $Q$ from a simpler family and observing the symmetries of the model, or by reparametrizing the model (see for example [1]). In what follows, we will develop a technique that uses the parts of the free energy to infer the mapping of the data to a class variable with an increased accuracy despite possible imperfections of the data fit, whether this imperfection is due to the approximations and errors in the model or the posterior.

Having obtained an estimate of parameters $\hat{\theta}$ that fit the given i.i.d. data we can rearrange the free energy (Eq.2) as

$$\mathcal{F}_\mathcal{Q} = \sum_t \mathcal{F}_\mathcal{Q}^t, \quad \text{and}$$

$$\mathcal{F}_\mathcal{Q}^t = \sum_{h^{(t)}} q(h^{(t)}|\hat{\theta}) \cdot \log q(h^{(t)}|\hat{\theta}) - \sum_{h^{(t)}} q(h^{(t)}|\hat{\theta}) \cdot \log P(h^{(t)}, x^{(t)}|\hat{\theta}) \tag{3}$$

The second term in the equation above is the *cross-entropy* term and it quantifies how well the data point fits the model, assuming that hidden variables follow the estimated posterior distribution. This posterior distribution is fit to minimize the free energy; the first term in 3 is the *entropy* and quantifies the uncertainty in this fit.

If $Q$ and $P$ factorize, then each of these two terms further breaks into a sum of individual terms, each quantifying the aspects of the fit of the data point with respect to different parts of the model. For example, if the generative model is described by a Bayesian network, the joint distribution can be written as $P(v^{(t)} = \prod_n P(v_n^{(t)}|\mathbf{PA}_n)$, where $v^{(t)} = \{x^{(t)}, h^{(t)}\}$ denotes the set of all variables (hidden or visible) and $\mathbf{PA}_n$ are the parents of the $n - th$ of these variables, i.e., $v_n^{(t)}$.

The cross-entropy term in the equation above further decomposes into

$$\sum_{[v_1^{(t)}]} q(v_1^{(t)} \cup \mathbf{PA}_1|\hat{\theta}) \cdot \log P(v_1^{(t)}|\mathbf{PA}_1, \hat{\theta}) + \cdots + \sum_{[v_N^{(t)}]} q(v_N^{(t)} \cup \mathbf{PA}_N|\hat{\theta}) \cdot \log P(v_N^{(t)}|\mathbf{PA}_N, \hat{\theta}) \tag{4}$$

For each discrete hidden variable $v_n^{(t)}$, the appropriate terms above can be further broken down into individual terms in the summation over the $D_n$ possible configurations of the variable, e.g.,

$$q(v_n^{(t)} = 1, \cup \mathbf{PA}_n|\hat{\theta}) \cdot \log P(v_n^{(t)} = 1|\mathbf{PA}_n, \hat{\theta}) + \cdots + q(v_n^{(t)} = D_n, \cup \mathbf{PA}_n|\hat{\theta}) \cdot \log P(v_n^{(t)} = D_n|\mathbf{PA}_n, \hat{\theta}) \tag{5}$$

In a similar fashion, the entropy term can also be decomposed further into a sum of terms as dictated by the factorization of the family $\mathcal{Q}$. Therefore, the free energy for a single sample $t$ can be expressed as the sum

$$\mathcal{F}_\mathcal{Q}^t = \sum_i f_{i,\hat{\theta}}^t \tag{6}$$

where all the free energy pieces $f_{i,\hat{\theta}}^t$ derive from the finest decomposition (5) or (4).

The terms $f_{i,\hat{\theta}}^t$ describe how the data point fits possible configurations of the hidden variables in different parts of the model. Such information can be encapsulated in a score space that we call *free energy score space* or simply **FESS**.

For example, in the case of a binary classification problem, given the generative models for the two classes, we can define as $\mathcal{F}_{(\mathcal{Q}, \hat{\theta})}(x^{(t)})$ the mapping of $x^{(t)}$ to a vector of scores $f$ with respect to a particular model with its estimated parameters, and a particular choice of the posterior family $\mathcal{Q}$ for each of the classes, and then concatenate the scores. Therefore, using the notation from [24] the free

energy score operator $\varphi_{\hat{F}}^{FESS}(x^{(t)})$ is defined as

$$\varphi_{\hat{F}}^{FESS} : x^{(t)} \rightarrow \left[ \mathcal{F}_{(\mathcal{Q}_1, \hat{\theta}_1)}(x^{(t)}); \mathcal{F}_{(\mathcal{Q}_2, \hat{\theta}_2)}(x^{(t)}) \right] \quad where \quad \mathcal{F}_{(\mathcal{Q}_c, \hat{\theta}_c)} = [\dots, f_{i,\hat{\theta}_c}^t, \dots]^T, c = 1, 2 \tag{7}$$

If the posterior families are fully expressive, then the MAP estimate based on the generative models for the two classes can be obtained from this mapping by simply summing the appropriate terms to obtain the log likelihood difference, as the free energy equals the negative log likelihood.

However, the mapping also allows for the parts of the model fit to play uneven roles in classification after an additional step of discriminative training. In this case the data points do not have to fit either model well in order to be correctly classified. Furthermore, even in the extreme case where one model provides a higher likelihood than the other for the data from *both* classes (e.g., because the models are not nested, and likelihoods cannot be directly compared), the mapping may still provide an abstraction from which another step of discriminative training can benefit. The additional step of training a discriminative model allows for mining the similarities among the data points in terms of the path through different hidden variables that has to be followed in their generation. These similarities may be informative even if the generative process is imperfect.

Obviously, (7) can be generalized to include multiple models (or the use of a single model) and/or multiple posterior approximations, either for two-class or multi-class classification problems.

## 3 Free energy score space classification dominates the MAP classification

We use here the terminology introduced in [27], under which **FESS** would be considered a *model-dependent feature extractor*, as different generative models lead to different feature vectors [25]. The family of feature extractors $\varphi_{\hat{F}} : \mathcal{X} \rightarrow \Re^d$ maps the input data $x \in \mathcal{X}$ in a space of fixed dimension derived from a plug-in estimate $\lambda$, in our case the generative model with parameters $\hat{\theta}$ from which the features are extracted.

Given some observations $x$ and the corresponding class labels $y \in \{-1, +1\}$ following the joint probability $P(x, y|\theta^*)$, a generative model can be trained to provide an estimate $\hat{\theta} \neq \theta^*$, where $\theta^*$ are the true parameters. As most kernels (e.g. Fisher and TOP) are commonly used in combination with linear classifiers such as linear SVMs, [27] proposes as a starting point for evaluating the performance of a feature extractor the classification error of a linear classifier $w^T \cdot \varphi_{\hat{F}}(x) + b$ in the feature space $\Re^d$, where $w \in \Re^d$ and $b \in \Re$. Assuming that $w$ and $b$ are chosen by an optimal learning algorithm on a sufficiently large training dataset, and that the test set follows the same distribution with parameter $\theta^*$, the classification error $R(\varphi_{\hat{F}})$ can be shown to tend to

$$R(\varphi_{\hat{F}}) = \min_{w,b} E_{x,y} \Phi[-y(w^T \cdot \varphi_{\hat{F}}(x) + b)] \tag{8}$$

where $\Phi[a]$ is an indicator function which is 1 when $a > 0$, and 0 otherwise, and $E_{x,y}$ denotes the expectation with respect to the true distribution $P(x, y|\theta^*)$.

The Fisher kernel (FK) classifier can perform at least as well as its plug-in estimate if the parameters of a linear classifier are properly determined [9, 27],

$$R(\varphi_{\hat{F}}^{FK}) \leq E_{x,t} \Phi[-y(P(y = +1|x, \hat{\theta}) - \frac{1}{2})] = R(\lambda) \tag{9}$$

where $\lambda$ represents the generative model used as plug-in estimate.

This property also trivially holds for our method, where $\varphi_{\hat{F}}(x^{(t)}) = \varphi_{\hat{F}}^{FESS}(x^{(t)})$, because the free energy can be expressed as a linear combination of the elements of $\varphi$.

In fact, the minimum free energy test (and the maximum likelihood rule when $\mathcal{Q}$ is fully expressive) can be defined on $\varphi$ derived from the generative models with parameters $\hat{\theta}_{+1}$ for one class and $\hat{\theta}_{-1}$ for another as

$$\hat{y} = \min_y \{\mathcal{F}_{(\mathcal{Q}, \hat{\theta}_{+1})}^t, \mathcal{F}_{(\mathcal{Q}, \hat{\theta}_{-1})}^t\} = \Phi\left[ \mathbf{1}^T \mathcal{F}_{(\mathcal{Q}, \hat{\theta}_{+1})}(x^{(t)}) - \mathbf{1}^T \mathcal{F}_{(\mathcal{Q}, \hat{\theta}_{-1})}(x^{(t)}) \right] \tag{10}$$

The extension to a multiclass classification is straightforward. When the family $\mathcal{Q}$ is expressive enough to capture the true posterior distribution, then free energy reduces to negative log likelihood,

and the free energy test reduces to ML classification. In other cases, likelihood computation is intractable, and free energy test is used instead of the likelihood ratio test. It is straightforward to prove that a kernel classifier that works in **FESS** is asymptotically at least as good as the MAP labelling based on the generative models for the two classes since generative classification is a special case of our framework.

**Lemma 3.1** *For $\varphi_{\hat{F}}^{FESS}(x^{(t)})$ derived as above with its first $M_1$ elements being the components of the free energy for one model, and the remaining $M_2$ for the second, a linear classifier employing $\varphi_{\hat{F}}^{FESS}$ will, asymptotically (with enough data), provide classification error which is at least as low as $R_{\mathcal{Q}}(\lambda)$ achieved using the free energy test above.*

$$R(\varphi_{\hat{F}}^{FESS}) \leq E_{x,t}\Phi\left[-y(P(y=+1|x,\hat{\theta}) - \frac{1}{2})\right] = R_{\mathcal{Q}}(\lambda)$$

**Proof**

$$R(\varphi_{\hat{F}}^{FESS}) = \min_{w,b} E_{x,y}\Phi[-y(w^T \cdot \varphi_{\hat{F}}^{FESS}(x) + b)] \leq E_{x,y}\Phi[-y(w^T \cdot \varphi_{\hat{F}}^{FESS}(x) + b)] \ \forall \ w,b$$

$$R(\varphi_{\hat{F}}^{FESS}) \leq E_{x,y}\Phi[-y(w_g^T \cdot \varphi_{\hat{F}}^{FESS}(x) + b_g)] \ for \ w_g = [\overbrace{+1,\cdots,+1}^{M_1 \ times}, \overbrace{-1,\cdots,-1}^{M_2 \ times}]^T, \ b_g = 0$$

$$R(\varphi_{\hat{F}}^{FESS}) \leq R_{\mathcal{Q}}(\lambda) \tag{11}$$

$\square$

Furthermore, when the family $\mathcal{Q}$ is expressive enough to capture the true posterior distribution, the free energy test is equivalent to maximum likelihood (ML) classification, $R_{\mathcal{Q}}(\lambda) = R(\lambda)$. The dominance of the Fisher and Top kernels [9, 27] over their plug-in holds for **FESS** too, and the same plug-in (the likelihood under a generative model) may be used when this is tractable. However, if the computation of the likelihood (and the kernels derived from it) is intractable, then the free energy test as well as the kernel methods based on **FESS** that will outperform this test, can both still be used with an appropriate family of variational distributions $\mathcal{Q}$.

## 4 Controlling the length of the feature vector

In some generative models, especially sequence models, the number of hidden variables may change from one data point to the next. In speech processing, for instance, hidden Markov models (HMM) [23] may have to model utterances $x_1^{(t)}, \ldots, x_{K(t)}^{(t)}$ of different sequence lengths $K(t)$. As each element in the sequence has an associated hidden variable, the hidden state sequences $s_1^{(t)}, \ldots, s_{K(t)}^{(t)}$ are also of variable lengths. The parameters $\theta$ of this model include the prior state distribution $\pi$, the state transition probability matrix $\mathbf{A} = a_{\{ij\}}$, and the emission probabilities $\mathbf{B} = b_{\{iv\}}$. Exact inference is tractable in HMMs and so we can use the exact posterior (EX) distribution to formulate the free energy and the free energy minimization is equivalent to the usual Baum-Welch training algorithm [17] and $\mathcal{F}_{EX} = -\log P(x)$. The free energy of each sample $x^t$ is

$$\mathcal{F}_{EX}^t = \sum_{[s]} q(s_1^{(t)})\log q(s_1^{(t)}) + \sum_{[s]}\sum_{k=1}^{K(t)-1} q(s_k^{(t)}, s_{k+1}^{(t)})\log q(s_k^{(t)}, s_{k+1}^{(t)}) - \sum_{[s]} q(s_1^{(t)})\log \pi_{s_1^{(t)}}$$

$$- \sum_{[s]}\sum_{k=1}^{K(t)-1} q(s_k^{(t)}, s_{k+1}^{(t)})\log a_{\{s_k^{(t)}, s_{k+1}^{(t)}\}} - \sum_{[s]}\sum_{k=1}^{K(t)} q(s_k^{(t)})\log b_{\{s_k^{(t)}, x_k^{(t)}\}} \tag{12}$$

Depending on how this is broken into terms $f_i$, we could get feature vectors whose dimension depends on the length of the sample $K(t)$. To solve this problem, we first note that a standard approach to dealing with utterances of different lengths is to normalize the likelihood by the sequence length, and this approach is also used for defining other score spaces. If, before the application of the score operator, we simply evaluate the sums over $k$ in the free energy and divide each by $K(t)$, we obtain a fixed number of terms independent of the sequence length. This results in a length-normalized score space **nFESS**, where the granularity of the decomposition of the free energy is dramatically reduced.

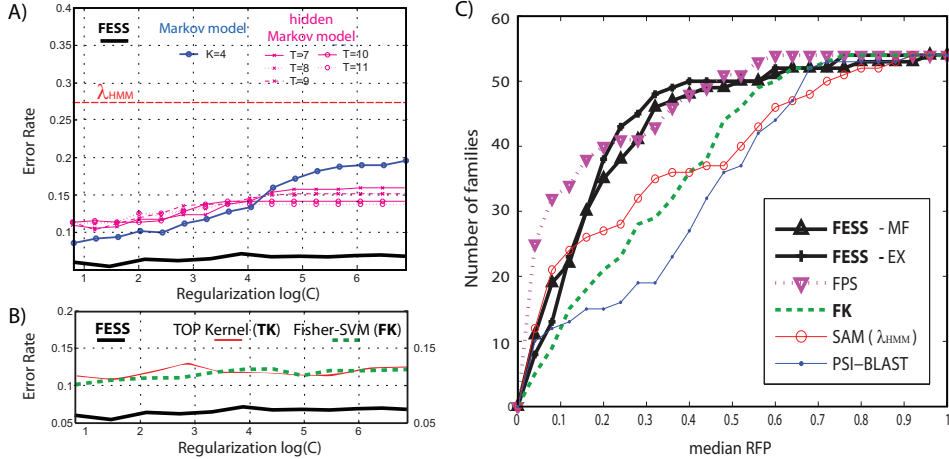

Figure 1: A) SVM error rates for **nFESS** and probability product kernels [10] using Markov models (we reported only their best result) and hidden Markov models as plug-ins. $T$ represents the parameters used in the kernel of [10], and $K$ is the order of the Markov chain. The results are arranged along the x axis by the regularization constant used in SVM training. B) Comparison with results obtained using **FK** and **TK** score spaces. C) Comparison of the five homology detection methods in Experiment 3. Y axis represents the total number of families for which a given method exceeds a median RFP score on the X axis.

In general, even for fixed-length data points and arbitrary generative models, we do not need to create large feature vectors corresponding to the finest level of granularity described in (5), or for that matter the slightly coarser level of granularity in (4). Some of the terms in these equations can be grouped and summed up to ensure for shorter feature vectors, if this is warranted by the application. The longer the feature vector, the finer is the level of detail with which the generative process for the data sample is represented, but more data is needed for the training of the discriminative classifier. Domain knowledge can often be used to reduce the complexity of the representation by summing appropriate terms without sacrificing the amount of useful information packed in the feature vectors.

Such control of the feature vector length does not negate the previously discussed advantages of the classification in the free energy score space compared with the straightforward application of free energy, likelihood, or in case of sequence models, length-normalized likelihood tests.

## 5 Experiments

We evaluated our approach on four standard datasets and compared its performance with the classification results provided by the datasets' creators, those estimated using the plug-in estimate $\lambda$, and those obtained using the Fisher (**FK**) and TOP (**TK**) kernel [9, 27] derived from the plug-ins. Support vector machines (SVMs) with RBF kernel were used as discriminative classifiers in all the score spaces, as this technique was previously identified as most potent for dealing with variable-length sequences [25]. As plug-ins, or generative models/likelihoods $\lambda$, for the three score spaces we compare across experiments, we used hidden Markov models (HMMs)[23] in Experiments 1-3 and latent Dirichlet allocation (LDA)[4] in Experiment 4. For each experiment, comparisons are based on the same validation procedure used in the appropriate original papers that introduced the datasets. For both **FK** and **FESS**, in each experiment we trained a single generative model (HMM or LDA, depending on the experiment). For all HMM models, the length-normalization with associated summation over the sequence as described in the previous section was used in the construction of the free energy score space. The model complexity, e.g., the number of states for the HMM were chosen by cross-validation on the training set.

**Experiment 1: E. coli promoter gene sequences.** The first analyzed dataset consists of the E. coli promoter gene sequences (DNA) with associated imperfect domain theory [26]. The standard task on this dataset is to recognize promoters in strings of nucleotides (A, G, T, or C). A promoter is a genetic region which facilitates the transcription of gene located nearby. The input features are 57 sequential DNA nucleotides. Results, obtained using leave-one-out (LOO) validation, are

reported in Table 1 and illustrate that **FESS** represents well the fixed size genetic sequences, leading to superior performance over other score spaces as well as over the plug-in $\lambda_{HMM}$.

| E.Coli | $\lambda_{HMM}$ | FESS | nFESS | FK | TK |
|---|---|---|---|---|---|
| Accuracy | 67,34% | 94,33% | 85,80% | 79,20% | 85,30% |

Table 1: Promoter classification results.

**Experiment 2: Introns/Exons classification in HS³D data set.** The $HS^3D$ data set [1][10] contains labelled intron and exon sequences of nucleotides. The task here is to distinguish between the two types of gene sequences that can both vary in length (from dozens of nucleotides to tens of thousands of nucleotides). For the sake of comparison, we adopted the same experimental setting of [10]. In Fig.1-A (top right), we reported the results obtained in [10] (overall error rate, OER, 7,5%), the results obtained using the HMM model ($\lambda_{HMM}$, OER 27,59%) together with the results obtained by our method (OER 6,12%). In Fig. 1-B (bottom right), we compared our method also with **FK** (OER 10,06%) and **TK** (OER 12,82%) kernels.

**Experiment 3: Homology detection in SCOP 1.53.** We tested the ability of **FESS** to classify protein domains into superfamilies in the Structural Classification of Proteins (SCOP)[2] version 1.53. The sequences in the database were selected from the Astral database, based on the E-value threshold of $10^{-25}$ for removing similar sequences from it. In the end, 4352 distinct sequences were grouped into families and superfamilies. For each family, the protein domains within the family are considered positive test examples, and the protein domains outside the family, but within the same superfamily, are taken as positive training examples. The data set yields 54 families containing at least 10 family members (positive test) and 5 superfamily members outside of the family (positive train) for a total of 54 One-Vs-All problems. The experimental setup is similar to that used in [8], except for one important difference: in the current experiments, the positive training sets do not include additional protein sequences extracted from a large, unlabelled database. Therefore, the recognition tasks performed here are more difficult than those in [8]. In order to measure the quality of the ranking, we used the median RFP score [8] which is the fraction of negative test sequences that score as high as or better than the median-scoring positive sequence. We used SVM decision values as score. We find that **FESS** outperforms task-specific algorithms (PSI-Blast [2] and SAM [14]) as well as the Fisher score (**FK**,[8]) with statistical significance with *p-values* of 5.1e-9, 8.3e-7, 1.1e-5, respectively. There is no statistical difference between our results **FESS** and those based on FPS [3]. In particular, the poor performance of [8] is explained by the under-training of HMMs [6]. The **FESS** representation proved to be much less sensitive to the training problems. We repeated the test using two different choices of $\mathcal{Q}$: the approximate mean field factorization and the exact posterior (**FESS**-MF and **FESS**-EX, respectively, in Fig.1-C). Interestingly, the performance was also robust with respect to these choices.

**Experiment 4: Scene/object recognition.** Our final set of experiments used the data from the Graz dataset[3], as well as the dataset proposed in [21]. In both tests, we used Latent Dirichlet allocation (LDA) [4] as the generative model. The free energy for LDA is derived in [4]. To serve as words in the model, we extracted SIFT features from 16x16 pixel windows computed over a grid with spacing of 8 pixels. These features were mapped to 175 codewords ($W = 175$). We varied the number of topics to explore the effectiveness of different techniques.

Graz dataset has two object classes, bikes (373 images) and persons (460 images), in addition to a background class (270 images)[4]. The range of scales and poses at which exemplars are presented is highly diverse, e.g., a "person" image may show a pedestrian at a certain distance, a side view of a complete body, or just a closeup of a head. We performed two-class detection (object vs. background) using an experimental setup consistent with [16, 22]. We generated ROC curves by thresholding raw SVM output, and report here the ROC equal error rate averaged over ten runs. The results are shown in Table 2. The standard deviation of the classification rate is quite high as the images in the database have very different complexities, and the performance for any single run is

| Graz dataset | FESS - Z=15 | FESS - Z=30 | FESS - Z=45 | [16] | [22] |
|---|---|---|---|---|---|
| Bikes | 86,1% (1,8) | 86,5% (2,0) | **89,1%** (2,3) | 86,3% (2,5) | 86,5% |
| People | 83,1% (3,1) | 82,9% (2,8) | **84,4%** (2,0) | 82,3% (3,1) | 80,8% |
| **Scenes dataset** | $\lambda_{LDA}$ | **FESS** | **FK** | [21] | [16] |
| Natural | 63,93% | **95,21%** | 90,10% | 89,00% | 84,51% |
| Artificial | 67,21% | **94,38%** | 90,32% | 89,00% | 89,43% |

Table 2: Classification rates for object/scene recognition tasks. The deviation is shown in brackets. Our approach tends to be robust to the choice of the number $Z$ of topics, and so in scene recognition experiments, we report only the result for Z=40.

highly dependent on the composition of the training set.

We also tested our approach on the scene recognition task using the datasets of [21], composed of two (Natural and Artificial scenes) datasets, each with 4 different classes. The results are reported in Table 2 where for the first time we employed Fisher-LDA in a vision application. Although this new technique outperformed state of the art, once again, **FESS** outperforms both this result and other state-of-the-art discriminative methods [21, 16].

## 6    Conclusions

In this paper, we present a novel generative score space, **FESS**, exploiting variational free energy terms as features. The additive free energy terms arise naturally as a consequence of the factorization of the model $P$ and the posterior $Q$. We show that the use of these terms as features in discriminative classification leads to more robust results than the use of the Fisher scores, which are based on the derivatives of the log likelihood of the data with respect to the model parameters. As was previously observed, we find that the Fisher score space suffers from the so called "wrap-around" problem, where very different data points may map to the same derivative, an example of which was discussed in the introduction. The free energy terms, on the other hand, quantify the data fit in different parts of the model, and seem to be informative even when the model is imperfect. This indicates that the re-scaling of these terms, which the subsequent discriminative training provides, leads to improved modelling of the data in some way. Scaling a term in the free energy composition, e.g., the term $\sum_h q(h) \log p(x|h)$, by a constant $w$ is equivalent to raising the appropriate conditional distribution to the power $w$. This is indeed reminiscent of some previous approaches to correcting generative modelling problems. In speech applications, for example, it is a standard practice to raise the observation likelihood in HMMs to a power less than 1, before inference is performed on the test sample, as the acoustic signal would otherwise overwhelm the hidden process modelling the language constraints [28]. This problem arises from the approximations in the acoustic model. For instance, a high-dimensional acoustic observation is often modelled as following a diagonal Gaussian distribution, thus assuming independent noise in the elements of the signal, even though the true acoustics of speech is far more constrained. This results in over-accounting for the variation in the observed acoustic signal, and to correct for this in practice, the log probability of the observation given the hidden variable is scaled down. The technique described here proposes a way to automatically infer the best scaling, but it also goes a step further in allowing for such corrections at all levels of the model hierarchy, and even for specific configurations of hidden variables. Furthermore, the use of kernel methods provides for nonlinear corrections, as well. This extremely simple technique was shown here to work remarkably well, outperforming previous score space approaches as well as the state of the art in multiple applications.

It is possible to extend the ideas here to other types of model/data energy. For example, the free energy approximated in different ways is used in [1] to construct various inference algorithms for a single scene parsing task. It may also be effective, for example, to use the terms in the Bethe free energy linked to different belief propagation messages to construct the feature vectors. Finally, although we find that **FESS** outperforms the previously studied score spaces that depend on the derivatives, i.e. where $\hat{F}$ is a derivative with respect to $\theta$, the use of this derivative in (7) is, of course, possible. This allows for the construction of kernels similar to **FK** and **TK**, but derived from intractable generative models as we show in Experiment 4 (**FK** in Table 2) on latent Dirichlet allocation.

## Acknowledgements

We acknowledge financial support from the FET programme within the EU FP7, under the SIMBAD project (contract 213250).

## Footnotes

[1] www.sci.unisannio.it/docenti/rampone

[2] http://scop.mrc-lmb.cam.ac.uk/scop/

[3] http://www.emt.tugraz.at/ pinz/data/GRAZ_02/

[4] The car class is ignored as in [16]

# References

[1] B. Frey and N. Jojic. A Comparison of Algorithms for Inference and Learning in Probabilistic Graphical Models *Transactions on pattern analysis and machine intelligence*, 1392:1416–27, 2005.

[2] S. F. Altschul, W. Gish, W. Miller, E. W. Myers, and D. J. Lipman. Basic local alignment search tool. *J Mol Biol*, 215(3):403–410, October 1990.

[3] T. L. Bailey and W. N. Grundy. Classifying proteins by family using the product of correlated p-values. In *Proceedings of the Third Annual International Conference on Computational Molecular Biology*, pages 10–14. ACM, 1999.

[4] D. M. Blei, A. Y. Ng, and M. I. Jordan. Latent dirichlet allocation. *J. Mach. Learn. Res.*, 3:993–1022, 2003.

[5] G. Bouchard and B. Triggs. The tradeoff between generative and discriminative classifiers. In *IASC International Symposium on Computational Statistics*, pages 721–728, Prague, August 2004.

[6] K. Tsuda B.Schlkopf and J. Vert. *Kernel Methods in Computational Biology*. The MIT Press, 2004.

[7] Z. Ghahramani. On structured variational approximations. Technical Report CRG-TR-97-1, 1997.

[8] T. Jaakkola, M. Diekhaus, and D. Haussler. Using the fisher kernel method to detect remote protein homologies. *7th Intell. Sys. Mol. Biol.*, pages 149–158, 1999.

[9] T. Jaakkola and D. Haussler. Exploiting generative models in discriminative classifiers. *Nips*, 1998.

[10] T. Jebara, R. Kondor, A. Howard, K. Bennett, and N. Cesa-bianchi. Probability product kernels. *Journal of Machine Learning Research*, 5:819–844, 2004.

[11] M.I. Jordan, Z. Ghahramani, T. Jaakkola, and L.K. Saul. An introduction to variational methods for graphical models. *Machine Learning*, 37(2):183–233, 1999.

[12] S. Kapadia. *Discriminative Training of Hidden Markov Models*. PhD thesis, 1998.

[13] H. Kappen and W. Wiegerinck. Mean field theory for graphical models, 2001.

[14] K. Karplus, C. Barrett, and R. Hughey. Hidden markov models for detecting remote protein homologies. *Bioinformatics*, 14:846–856, 1999.

[15] J. A. Lasserre, C. M. Bishop, and T. P. Minka. Principled hybrids of generative and discriminative models. In *Cvpr*, pages 87–94, 2006.

[16] S. Lazebnik, C. Schmid, and J. Ponce. Beyond bags of features: Spatial pyramid matching for recognizing natural scene categories. *Cvpr*, 2:2169–2178, 2006.

[17] D. MacKay. Ensemble learning for Hidden Markov Models, 1997. Unpublished. Department of Physics, University of Cambridge.

[18] A. Mccallum, C. Pal, G. Druck, and X. Wang. Multi-conditional learning: Generative/discriminative training for clustering and classification. In *In Proceedings of the 21st National Conference on Artificial Intelligence*, pages 433–439, 2006.

[19] R. M. Neal and G. E. Hinton. A view of the em algorithm that justifies incremental, sparse, and other variants. pages 355–368, 1999.

[20] A. Y. Ng and M. I. Jordan. On discriminative vs. generative classifiers: A comparison of logistic regression and naive bayes. In T. G. Dietterich, S. Becker, and Z. Ghahramani, editors, *NIPS*, Cambridge, MA, 2002. MIT Press.

[21] A. Oliva and A. Torralba. Modeling the shape of the scene: A holistic representation of the spatial envelope. *International Journal of Computer Vision*, 42:145–175, 2001.

[22] A. Opelt, M. Fussenegger, A. Pinz, and P. Auer. Weak hypotheses and boosting for generic object detection and recognition. In *Eccv*, volume 2, pages 71–84, 2004.

[23] L.R. Rabiner. A tutorial on Hidden Markov Models and selected applications in speech recognition. *Proc. of IEEE*, 77(2):257–286, 1989.

[24] N. Smith and M. Gales. Speech recognition using SVMs. In *Nips*, pages 1197–1204. MIT Press, 2002.

[25] N. Smith and M. Gales. Using svms to classify variable length speech patterns. Technical Report CUED/F-INGENF/TR.412, University of Cambridge, UK, 2002.

[26] G. G. Towell, J. W. Shavlik, and M. O. Noordewier. Refinement of approximate domain theories by knowledge-based neural networks. In *In Proceedings of the Eighth National Conference on Artificial Intelligence*, pages 861–866, 1990.

[27] K. Tsuda, M. Kawanabe, G. Rätsch, S. Sonnenburg, and K. R. Müller. A new discriminative kernel from probabilistic models. *Neural Comput.*, 14(10):2397–2414, 2002.

[28] L. Deng and D. O'Shaughnessy, Speech Processing - A Dynamic and Optimization-Oriented Approach *Marcel Dekker Inc.*, June 2003

